# Scalable Algorithms for String Kernels with Inexact Matching

**Pavel P. Kuksa, Pai-Hsi Huang, Vladimir Pavlovic**
Department of Computer Science,
Rutgers University, Piscataway, NJ 08854
{pkuksa,paihuang,vladimir}@cs.rutgers.edu

## Abstract

We present a new family of linear time algorithms for string comparison with mismatches under the string kernels framework. Based on sufficient statistics, our algorithms improve theoretical complexity bounds of existing approaches while scaling well in sequence alphabet size, the number of allowed mismatches and the size of the dataset. In particular, on large alphabets and under loose mismatch constraints our algorithms are several orders of magnitude faster than the existing algorithms for string comparison under the mismatch similarity measure. We evaluate our algorithms on synthetic data and real applications in music genre classification, protein remote homology detection and protein fold prediction. The scalability of the algorithms allows us to consider complex sequence transformations, modeled using longer string features and larger numbers of mismatches, leading to a state-of-the-art performance with significantly reduced running times.

## 1   Introduction

Analysis of large scale sequential data has become an important task in machine learning and data mining, inspired by applications such as biological sequence analysis, text and audio mining. Classification of string data, sequences of discrete symbols, has attracted particular interest and has led to a number of new algorithms [1, 2, 3, 4]. They exhibit state-of-the-art performance on tasks such as protein superfamily and fold prediction, music genre classification and document topic elucidation.

Classification of data in sequential domains is made challenging by the variability in the sequence lengths, potential existence of important features on multiple scales, as well as the size of the alphabets and datasets. Typical alphabet sizes can vary widely, ranging in size from 4 nucleotides in DNA sequences, up to thousands of words from a language lexicon for text documents. Strings within the same class, such as the proteins in one fold or documents about politics, can exhibit wide variability in the primary sequence content. Moreover, important datasets continue to increase in size, easily reaching millions of sequences. As a consequence, the resulting algorithms need the ability to efficiently handle large alphabets and datasets as well as establish measures of similarity under complex sequence transformations in order to accurately classify the data.

A number of state-of-the-art approaches to scoring similarity between pairs of sequences in a database rely on fixed, spectral representations of sequential data and the notion of mismatch kernels, c.f. [2, 3]. In that framework an induced representation of a sequence is typically that of the spectra (counts) of all short substrings (k-mers) contained within a sequence. The similarity score is established by allowing transformations of the original k-mers based on different models of deletions, insertions and mutations. However, computing those representations efficiently for large alphabet sizes and "loose" similarity models can be computationally challenging. For instance, the complexity of an efficient trie-based computation [3, 5] of the mismatch kernel between two strings $X$ and $Y$ strongly depends on the alphabet size and the number of mismatches allowed as

$O(k^{m+1}|\Sigma|^m(|X| + |Y|))$ for $k$-mers (contiguous substring of length $k$) with up to $m$ mismatches and the alphabet size $|\Sigma|$. This limits the applicability of such algorithms to simpler transformation models (shorter $k$ and $m$) and smaller alphabets, reducing their practical utility on complex real data. As an alternative, more complex transformation models such as [2] lead to state-of-the-art predictive performance at the expense of increased computational effort.

In this work we propose novel algorithms for modeling sequences under complex transformations (such as multiple insertions, deletions, mutations) that exhibit state-of-the-art performance on a variety of distinct classification tasks. In particular, we present new algorithms for inexact (e.g. with mismatches) string comparison that improve currently known time bounds for such tasks and show orders-of-magnitude running time improvements. The algorithms rely on an efficient implicit computation of mismatch neighborhoods and $k$-mer statistic on sets of sequences. This leads to a mismatch kernel algorithm with complexity $O(c_{k,m}(|X| + |Y|))$, where $c_{k,m}$ is independent of the alphabet $\Sigma$. The algorithm can be easily generalized to other families of string kernels, such as the spectrum and gapped kernels [6], as well as to semi-supervised settings such as the neighborhood kernel of [7]. We demonstrate the benefits of our algorithms on many challenging classification problems, such as detecting homology (evolutionary similarity) of remotely related proteins, recognizing protein fold, and performing classification of music samples. The algorithms display state-of-the-art classification performance and run substantially faster than existing methods. Low computational complexity of our algorithms opens the possibility of analyzing very large datasets under both fully-supervised and semi-supervised setting with modest computational resources.

## 2   Related Work

Over the past decade, various methods were proposed to solve the string classification problem, including *generative*, such as HMMs, or *discriminative* approaches. Among the discriminative approaches, in many sequence analysis tasks, kernel-based [8] machine learning methods provide most accurate results [2, 3, 4, 6].

Sequence matching is frequently based on common co-occurrence of exact sub-patterns ($k$-mers, features), as in spectrum kernels [9] or substring kernels [10]. Inexact comparison in this framework is typically achieved using different families of mismatch [3] or profile [2] kernels. Both spectrum-$k$ and mismatch($k$,$m$) kernel directly extract string features based on the observed sequence, $X$. On the other hand, the profile kernel, proposed by Kuang et al. in [2], builds a profile [11] $P_X$ and uses a similar $|\Sigma|^k$-dimensional representation, derived from $P_X$. Constructing the profile for each sequence may not be practical in some application domains, since the size of the profile is dependent on the size of the alphabet set. While for bio-sequences $|\Sigma| = 4$ or 20, for music or text classification $|\Sigma|$ can potentially be very large, on the order of tens of thousands of symbols. In this case, a very simple semi-supervised learning method, the *sequence neighborhood kernel*, can be employed [7] as an alternative to lone $k$-mers with many mismatches.

The most efficient available trie-based algorithms [3, 5] for mismatch kernels have a strong dependency on the size of alphabet set and the number of allowed mismatches, both of which need to be restricted in practice to control the complexity of the algorithm. Under the trie-based framework, the list of $k$-mers extracted from given strings is traversed in a depth-first search with branches corresponding to all possible $\sigma \in \Sigma$. Each leaf node at depth $k$ corresponds to a particular $k$-mer feature (either exact or inexact instance of the observed exact string features) and contains a list of matching features from each string. The kernel matrix is updated at leaf nodes with corresponding counts. The complexity of the trie-based algorithm for mismatch kernel computation for two strings $X$ and $Y$ is $O(k^{m+1}|\Sigma|^m(|X| + |Y|))$ [3]. The algorithm complexity depends on the size of $\Sigma$ since during a trie traversal, possible substitutions are drawn from $\Sigma$ explicitly; consequently, to control the complexity of the algorithm we need to restrict the number of allowed mismatches ($m$), as well as the alphabet size ($|\Sigma|$). Such limitations hinder wide application of the powerful computational tool, as in biological sequence analysis, mutation, insertions and deletions frequently co-occur, hence establishing the need to relax the parameter $m$; on the other hand, restricting the size of the alphabet sets strongly limits applications of the mismatch model. While other efficient string algorithms exist, such as [6, 12] and the suffix-tree based algorithms in [10], they do not readily extend to the mismatch framework. In this study, we aim to extend the works presented in [6, 10] and close the existing gap in theoretical complexity between the mismatch and other fast string kernels.

# 3 Combinatorial Algorithm

In this section we will develop our first improved algorithm for kernel computations with mismatches, which serves as a starting point for our main algorithm in Section 4.

## 3.1 Spectrum and Mismatch Kernels Definition

Given a sequence $X$ with symbols from alphabet $\Sigma$ the *spectrum-k* kernel [9] and the *mismatch(k,m)* kernel [3] induce the following $|\Sigma|^k$-dimensional representation for the sequence:

$$\Phi(X) \quad = \quad \left( \sum_{\alpha \in X} I_m(\alpha, \gamma) \right)_{\gamma \in \Sigma^k}, \tag{1}$$

where $I_m(\alpha, \gamma) = 1$ if $\alpha \in N_{k,m}(\gamma)$, and $N_{k,m}(\gamma)$ is the *mutational neighborhood*, the set of all $k$-mers that differ from $\gamma$ by at most $m$ mismatches. Note that, by definition, for spectrum kernels, $m = 0$.

The mismatch kernel is then defined as

$$K(X, Y | k, m) = \sum_{\gamma \in \Sigma^k} c_m(\gamma | X) c_m(\gamma | Y), \tag{2}$$

where $c_m(\gamma | X) = \Sigma_{\alpha \in X} I_m(\gamma, \alpha)$ is the number of times a contiguous substring of length $k$ ($k$-mer) $\gamma$ occurs in $X$ with no more than $m$ mismatches.

## 3.2 Intersection-based Algorithm

Our first algorithm presents a novel way of performing local inexact string matching with the following key properties:

- a. *parameter independent*: the complexity is independent of $|\Sigma|$ and mismatch parameter $m$
- b. *in-place*: only uses $\min(2m, k) + 1$ extra space for an auxiliary look-up table
- c. *linear complexity*: in $k$, the length of the substring (as opposed to exponential $k^m$)

To develop our first algorithm, we first write the mismatch kernel (Equation 2) in an equivalent form:

$$K(X, Y | k, m) = \sum_{i_x=1}^{n_x-k+1} \sum_{i_y=1}^{n_y-k+1} \sum_{a \in \Sigma^k} I_m(a, x_{i_x:i_x+k-1}) I_m(a, y_{i_y:i_y+k-1}) \tag{3}$$

$$= \sum_{i_x=1}^{n_x-k+1} \sum_{i_y=1}^{n_y-k+1} |(N(x_{i_x:i_x+k-1}, m) \cap N(y_{i_y:i_y+k-1}, m)| \tag{4}$$

$$= \sum_{i_x=1}^{n_x-k+1} \sum_{i_y=1}^{n_y-k+1} \mathcal{I}(x_{i_x:i_x+k-1}, y_{i_y:i_y+k-1}) \tag{5}$$

where $\mathcal{I}(a, b)$ is the number of induced (neighboring) $k$-mers common between $a$, $b$ (i.e. $\mathcal{I}(a, b)$ is the size of intersection of mismatch neighborhoods of $a$ and $b$). The key observation here is that if we can compute $\mathcal{I}(a, b)$ efficiently then the kernel evaluation problem reduces to performing pairwise comparison based on *all pairs of* observed $k$-mers, $a$ and $b$, in the two sequences. The complexity for such procedure is $O(c|X||Y|)$ where $c$ is the cost for evaluating $\mathcal{I}(a, b)$ for any given $k$-mers $a$ and $b$. In fact, for fixed $k, m$ and $\Sigma$, such quantity depends only on the Hamming distance $d(a, b)$ (i.e. the number of mismatches) and can be evaluated in advance, as we will show in Section 3.3. As a result, the intersection values can be looked up in a table in constant time during matching. Note the summation now shows no explicit dependency on $|\Sigma|$ and $m$. In summary, given two strings $X$ and $Y$, the algorithm (Algorithm 1) compares pairs of *observed* $k$-mers from $X$ and $Y$ and computes the mismatch kernel according to Equation 5.

| Algorithm 1. (Hamming-Mismatch) Mismatch algorithm based on Hamming distance |
|---|
| *Input*: strings $X, Y, |X| = n_x, |Y| = n_y$, parameters $k, m$, lookup table $\mathcal{I}$ for intersection sizes |
| Evaluate kernel using Equation 5: |
| $K(X, Y|k, m) = \sum_{i_x=1}^{n_x-k+1} \sum_{i_y=1}^{n_y-k+1} \mathcal{I}(d(x_{i_x:i_x+k-1}, y_{i_y:i_y+k-1})|k, m)$ |
| where $\mathcal{I}(d)$ is the intersection size for distance $d$ |
| *Output*: Mismatch kernel value $K(X, Y|k, m)$ |

The overall complexity of the algorithm is $O(kn_x n_y)$ since the Hamming distances between all $k$-mer pairs observed in $X$ and $Y$ need to be known. In the following section, we discuss how to efficiently compute the size of the intersection.

## 3.3 Intersection Size: Closed Form Solution

The number of neighboring $k$-mers shared by two observed $k$-mers $a$ and $b$ can be directly computed, in a closed-form, from the Hamming distance $d(a, b)$ for fixed $k$ and $m$, requiring *no* explicit traversal of the $k$-mer space as in the case of trie-based computations. We first consider the case $a = b$ (*i.e.* $d(a, b) = 0$). The intersection size corresponds to the size of the $(k, m)$-mismatch neighborhood, *i.e.* $\mathcal{I}(a, b) = |N_{k,m}| = \sum_{i=0}^{m} \binom{k}{i}(|\Sigma| - 1)^i$. For higher values of Hamming distance $d$, the key observation is that for fixed $\Sigma$, $k$, and $m$, given any distance $d(a, b) = d$, $I(a, b)$ is also a constant, regardless of the mismatch positions. As a result, intersection values can always be pre-computed *once*, stored and looked up when necessary. To illustrate this, we show two examples for $m = 1, 2$:

$$\mathcal{I}(a, b) \atop (m = 1) = \begin{cases} |N_{k,m}|, d(a, b) = 0 \\ |\Sigma|, d(a, b) = 1 \\ 2, d(a, b) = 2 \end{cases} \qquad \mathcal{I}(a, b) \atop (m = 2) = \begin{cases} |N_{k,m}|, d(a, b) = 0 \\ 1 + k(|\Sigma| - 1) + (k - 1)(|\Sigma| - 1)^2, d(a, b) = 1 \\ 1 + 2(k - 1)(|\Sigma| - 1) + (|\Sigma| - 1)^2, d(a, b) = 2 \\ 6(|\Sigma| - 1), d(a, b) = 3 \\ \binom{4}{2}, d(a, b) = 4 \end{cases}$$

In general, the intersection size can be found in a weighted form $\sum_i w_i(|\Sigma| - 1)^i$ and can be pre-computed in constant time.

## 4 Mismatch Algorithm based on Sufficient Statistics

In this section, we further develop ideas from the previous section and present an improved mismatch algorithm that does not require pairwise comparison of the $k$-mers between two strings and dependes *linearly* on sequence length. The crucial observation is that in Equation 5, $\mathcal{I}(a, b)$ is non-zero only when $d(a, b) \leq 2m$. As a result, the kernel computed in Equation 5 is incremented only by $\min(2m, k) + 1$ distinct values, corresponding to $\min(2m, k) + 1$ possible intersection sizes. We then can re-write the equation in the following form:

$$K(X, Y|m, k) = \sum_{i_x=1}^{n_x-k+1} \sum_{i_y=1}^{n_y-k+1} \mathcal{I}(x_{i_x:i_x+k-1}, y_{i_y:i_y+k-1}) = \sum_{i=0}^{\min(2m,k)} M_i \mathcal{I}_i, \qquad (6)$$

where $\mathcal{I}_i$ is the size of the intersection of $k$-mer mutational neighborhood for Hamming distance $i$, and $M_i$, the number of observed $k$-mer pairs in $X$ and $Y$ having Hamming distance $i$. The problem of computing the kernel has been further reduced to a single summation. We have shown in Section 3.3 that given any $i$, we can compute $\mathcal{I}_i$ in advance. The crucial task now becomes computing the *sufficient statistics* $M_i$ efficiently. In the following, we will show how to compute the mismatch statistics $\{M_i\}$ in $O(c_{k,m}(n_x + n_y))$ time, where $c_{k,m}$ is a constant that does *not* depend on the alphabet size. We formulate the task of inferring matching statistics $\{M_i\}$ as the following auxiliary counting problem:

> *Mismatch Statistic Counting*: Given a set of $n$ $k$-mers from two strings $X$ and $Y$, for each Hamming distance $i = 0, 1, ..., \min(2m, k)$, output the number of $k$-mer pairs $(a, b), a \in X, b \in Y$ with $d(a, b) = i$.

In this problem it is not necessary to know the distance between each pair of $k$-mers; one only needs to know *the number of* pairs ($M_i$) at each distance $i$. We show next that the above problem of computing matching statistics can be solved in *linear* time (in the number $n$ of $k$-mers) using multiple rounds of counting sort as a sub-algorithm.

We first consider the problem of computing number of $k$-mers at distance 0, i.e. the number of exact matches. In this case, we can apply counting sort to order all $k$-mers lexicographically and find the number of exact matches by scanning the sorted list. The counting then requires linear $O(kn)$ time. Efficient direct computation of $M_i$ for any $i > 0$ is difficult (requires quadratic time); we take another approach and first compute inexact *cumulative mismatch statistics*, $C_i = M_i + \sum_{j=0}^{i-1} \binom{k-j}{i-j} M_j$, that overcount the number of $k$-mer pairs at a given distance $i$, as follows. Consider two $k$-mers $a$ and $b$. Pick $i$ positions and remove from the $k$-mers the symbols at the corresponding positions to obtain $(k-i)$-mers $a'$ and $b'$. The key observation is that $d(a', b') = 0 \Rightarrow d(a, b) \leq i$. As a result, given $n$ $k$-mers, we can compute the cumulative mismatch statistics $C_i$ in linear time using $\binom{k}{i}$ rounds of counting sort on $(k-i)$-mers. The *exact* mismatch statistics $M_i$ can then be obtained from $C_i$ by subtracting the exact counts to compensate for overcounting as follows:

$$M_i = C_i - \sum_{j=0}^{i-1} \binom{k-j}{i-j} M_j, \quad i = 0, \ldots, \min(\min(2m, k), k-1) \tag{7}$$

The last mismatch statistic $M_k$ can be computed by subtracting the preceding statistics $M_0, \ldots M_{k-1}$ from the total number of possible matches:

$$M_k = T - \sum_{j=0}^{k-1} M_j, \quad \text{where} \quad T = (n_x - k + 1)(n_y - k + 1). \tag{8}$$

Our algorithm for mismatch kernel computations based on sufficient statistics is summarized in Algorithm 2. The overall complexity of the algorithm is $O(nc_{k,m})$ with the constant $c_{k,m} = \sum_{l=0}^{\min(2m,k)} \binom{k}{l}(k-l)$, independent of the size of the alphabet set, and $\binom{k}{l}$ is the number of rounds of counting sort for evaluating the cumulative mismatch statistics $C_l$.

---

**Algorithm 2. (Mismatch-SS) Mismatch kernel algorithm based on Sufficient Statistics**

*Input*: strings $X, Y, |X| = n_x, |Y| = n_y$, parameters $k, m$, pre-computed intersection values $\mathcal{I}$
1. Compute $\min(2m, k)$ *cumulative* matching statistics, $C_i$, using counting sort
2. Compute *exact* matching statistics, $M_i$, using Equation 7
3. Evaluate kernel using Equation 6: $K(X, Y|m, k) = \sum_{i=0}^{\min(2m,k)} M_i \mathcal{I}_i$
*Output*: Mismatch kernel value $K(X, Y|k, m)$

---

## 5   Extensions

Our algorithmic approach can also be applied to a variety of existing string kernels, leading to very efficient and simple algorithms that could benefit many applications.

**Spectrum Kernels**. The spectrum kernel [9] in our notation is the first sufficient statistic $M_0$, i.e. $K(X, Y|k) = M_0$, which can be computed in $k$ rounds of counting sort (i.e. in $O(kn)$ time).

**Gapped Kernels**. The gapped kernels [6] measure similarity between strings $X$ and $Y$ based on the co-occurrence of gapped instances $g$, $|g| = k + m > k$ of $k$-long substrings:

$$K(X, Y|k, g) = \sum_{\gamma \in \Sigma^k} \Big( \sum_{g \in X, |g|=k+m} I(\gamma, g) \Big) \Big( \sum_{g \in Y, |g|=k+m} I(\gamma, g) \Big), \tag{9}$$

where $I(\gamma, g) = 1$ when $\gamma$ is a subsequence of $g$. Similar to the algorithmic approach for extracting cumulative mismatch statistics in Algorithm-2, to compute the gapped($g$,$k$) kernel, we perform a single round of counting sort over $k$-mers contained in the $g$-mers. This gives a very simple and efficient $O(\binom{g}{k}kn)$ time algorithm for gapped kernel computations.

**Wildcard kernels**. The wildcard($k$,$m$) kernel [6] in our notation is the sum of the cumulative statistics $K(X, Y|k, m) = \sum_{i=0}^{m} C_i$, i.e. can be computed in $\sum_{i=0}^{m} \binom{k}{i}$ rounds of counting sort, giving a simple and efficient $O(\sum_{i=0}^{m} \binom{k}{i}(k-i)n)$ algorithm.

**Spatial kernels**. The spatial($k,t,d$) kernel [13] can be computed by sorting $kt$-mers iteratively for every arrangement of $t$ $k$-mers spatially constrained by distance $d$.

**Neighborhood Kernels**. The *sequence neighborhood* kernels [7] proved to be a powerful tool in many sequence analysis tasks. The method uses the unlabeled data to form a set of neighbors for train/test sequences and measure similarity of two sequences $X$ and $Y$ using their neighborhoods:

$$K(X,Y) = \sum_{x \in N(X)} \sum_{y \in N(Y)} K(x,y) \tag{10}$$

where $N(X)$ is *the sequence neighborhood* that contains neighboring sequences from the unlabeled data set, including $X$ itself. Note the kernel value, if computed directly using Equation 10, will incur *quadratic* complexity in the size of the neighborhoods. Similar to the single string case, using our algorithmic approach, to compute the neighborhood kernel (over the string sets), we can *jointly* sort the observed $k$-mers in $N(X)$ and $N(Y)$ and apply the desired kernel evaluation method (spectrum, mismatch, or gapped). Under this setting, the neighborhood kernel can be evaluated in time *linear* to the neighborhood size. This leads to very efficient algorithms for computing sequence neighborhood kernels even for very large datasets, as we will show in the experimental section.

# 6 Evaluation

We study the performance of our algorithms, both in running time and predictive accuracy, on synthetic data and standard benchmark datasets for protein sequence analysis and music genre classification. The reduced running time requirements of our algorithms open the possibility to consider "looser" mismatch measures with larger $k$ and $m$. The results presented here demonstrate that such mismatch kernels with larger $(k, m)$ can lead to state-of-the-art predictive performance even when compared with more complex models such as [2].

We use three standard benchmark datasets to compare with previously published results: the SCOP dataset (7329 sequences with 2862 labeled) [7] for remote protein homology detection, the Ding-Dubchak dataset[1] (27 folds, 694 sequences) [14, 15] for protein fold recognition, and music genre data[2] (10 classes, 1000 sequences, $|\Sigma| = 1024$) [16] for multi-class genre prediction. For protein sequence classification under the semi-supervised setting, we also use the Protein Data Bank (PDB, $17,232$ sequences), the Swiss-Prot ($101,602$ sequences), and the non-redundant (NR) databases as the unlabeled datasets, following the setup of [17]. All experiments are performed on a *single* $2.8GHz$ CPU. The datasets used in our experiments and the suplementary data/code are available at `http://seqam.rutgers.edu/new-inexact/new-inexact.html`.

## 6.1 Running time analysis

We compare the running time of our algorithm on synthetic and real data with the trie-based computations. For *synthetic data*, we generate strings of length $n = 10^5$ over alphabets of different sizes and measure the running time of the trie-based and our sufficient statistics based algorithms for evaluating mismatch string kernel. Figure 1 shows relative running time $T_{trie}/T_{ss}$, in *logarithmic scale*, of the mismatch-trie and mismatch-SS as a function of the alphabet size. As can be seen from the plot, our algorithm demonstrates several orders of magnitude improvements, especially for large alphabet sizes.

Table 1 compares running times of our algorithm and the trie-based algorithm for different real dataset (proteins, DNA, text, music) for a single kernel entry (pair of strings) computation. We observe the speed improvements ranging from 100 to $10^6$ times depending on the alphabet size.

We also measure the running time for full 7329-by-7329 mismatch(5,2) kernel matrix computations for SCOP dataset under the supervised setting. The running time of our algorithm is 1525 seconds compared to 196052 seconds for the trie-based computations. The obtained speed-up of 128 times is as expected from the theoretical analysis (our algorithm performs 31 counting-sort iterations in total over 5-, 4-, 3-, 2-, and 1- mers, which gives the running time ratio of approximately 125 when compared to the trie-based complexity). We observe similar improvements under

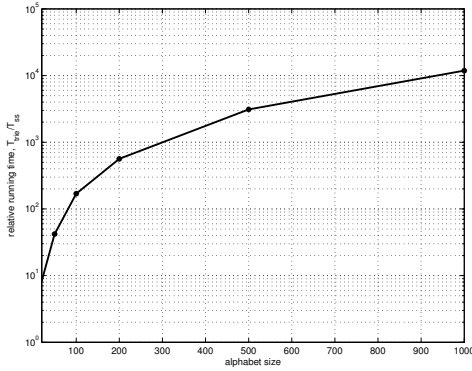

Figure 1: Relative running time $T_{trie}/T_{ss}$ (in logarithmic scale) of the mismatch-trie and mismatch-ss as a function of the alphabet size (mismatch(5,1) kernel, $n = 10^5$)

Table 1: Running time (in seconds) for kernel computation between two strings on real data

|  | long protein | protein | dna | text | music |
|---|---|---|---|---|---|
| n | 36672 | 116 | 570 | 242 | 6892 |
| $|\Sigma|$ | 20 | 20 | 4 | 29224 | 1024 |
| (5,1)-trie | 1.6268 | 0.0212 | 0.0260 | 20398 | 526.8 |
| (5,1)-ss | 0.1987 | 0.0052 | 0.0054 | 0.0178 | 0.0331 |
| time ratio | 8 | 4 | 5 | $10^6$ | 16,000 |
| (5,2)-trie | 31.5519 | 0.2918 | 0.4800 | - | - |
| (5,2)-ss | 0.2957 | 0.0067 | 0.0064 | 0.0649 | 0.0941 |
| time ratio | 100 | 44 | 75 | - | - |

the semi-supervised setting for neighborhood mismatch kernels; for example, computing a smaller neighborhood mismatch(5,2) kernel matrix for the *labeled sequences* only (2862-by-2862 matrix) using the Swiss-Prot unlabeled dataset takes $1,480$ seconds with our algorithm, whereas performing the same task with the trie-based algorithm takes about $5$ days.

## 6.2 Empirical performance analysis

In this section we show predictive performance results for several sequence analysis tasks using our new algorithms. We consider the tasks of the multi-class music genre classification [16], with results in Table 2, and the protein remote homology (superfamily) prediction [9, 2, 18] in Table 3. We also include preliminary results for multi-class fold prediction [14, 15] in Table 4.

On the music classification task, we observe significant improvements in accuracy for larger number of mismatches. The obtained error rate ($35.6\%$) on this dataset compares well with the state-of-the-art results based on the same signal representation in [16]. The remote protein homology detection, as evident from Table 3, clearly benefits from larger number of allowed mismatches because the remotely related proteins are likely to be separated by multiple mutations or insertions/deletions. For example, we observe improvement in the average ROC-50 score from 41.92 to 52.00 under a fully-supervised setting, and similar significant improvements in the semi-supervised settings. In particular, the result on the Swiss-Prot dataset for the $(7, 3)$-mismatch kernel is very promising and compares well with the best results of the state-of-the-art, but computationally more demanding, profile kernels [2]. The neighborhood kernels proposed by Weston et al. have already shown very promising results in [7], though slightly worse than the profile kernel. However, using our new algorithm that significantly improves the speed of the neighborhood kernels, we show that with larger number of allowed mismatches the neighborhood can perform even better than the state-of-the-art profile kernel: the (7,3)-mismatch neighborhood achieves the average ROC-50 score of 86.32, compared to 84.00 of the profile kernel on the Swiss-Prot dataset. This is an important result that addresses a main drawback of the neighborhood kernels, the running time [7, 2].

Table 2: Classification performance on music genre classification (multi-class)

| Method | Error |
|---|---|
| Mismatch (5,1) | 42.6±6.34 |
| Mismatch (5,2) | 35.6±4.99 |

Table 3: Classification performance on protein remote homology prediction

| dataset | mismatch (5,1) | | mismatch (5,2) | | mismatch (7,3) | |
|---|---|---|---|---|---|---|
|  | ROC | ROC50 | ROC | ROC50 | ROC | ROC50 |
| SCOP (supervised) | 87.75 | 41.92 | 90.67 | 49.09 | 91.31 | 52.00 |
| SCOP (unlabeled) | 90.93 | 67.20 | 91.42 | 69.35 | 92.27 | 73.29 |
| SCOP (PDB) | 97.06 | 80.39 | 97.24 | 81.35 | 97.93 | 84.56 |
| SCOP (Swiss-Prot) | 96.73 | 81.05 | 97.05 | 82.25 | 97.78 | 86.32 |

For multi-class protein fold recognition (Table 4), we similarly observe improvements in performance for larger numbers of allowed mismatches. The balanced error of $25\%$ for the (7,3)-mismatch neighborhood kernel using Swiss-Prot compares well with the best error rate of $26.5\%$ for the state-

of-the-art profile kernel with adaptive codes in [15] that used a much larger non-redundant (NR) dataset. Using NR, the balanced error further reduces to 22.5% for the (7,3)-mismatch.

Table 4: Classification performance on fold prediction (multi-class)

| Method | Error | Top 5 Error | Balanced Error | Top 5 Balanced Error | Recall | Top 5 Recall | Precision | Top 5 Precision | F1 | Top5 F1 |
|---|---|---|---|---|---|---|---|---|---|---|
| Mismatch $(5,1)$ | 51.17 | 22.72 | 53.22 | 28.86 | 46.78 | 71.14 | 90.52 | 95.25 | 61.68 | 81.45 |
| Mismatch $(5,2)$ | 42.30 | 19.32 | 44.89 | 22.66 | 55.11 | 77.34 | 67.36 | 84.77 | 60.62 | 80.89 |
| Mismatch $(5,2)^\dagger$ | 27.42 | 14.36 | 24.98 | 13.36 | 75.02 | 86.64 | 79.01 | 91.02 | 76.96 | 88.78 |
| Mismatch $(7,3)$ | 43.60 | 19.06 | 47.13 | 22.76 | 52.87 | 77.24 | 84.65 | 91.95 | 65.09 | 83.96 |
| Mismatch $(7,3)^\dagger$ | 26.11 | 12.53 | 25.01 | 12.57 | 74.99 | 87.43 | 85.00 | 92.78 | 79.68 | 90.02 |
| Mismatch $(7,3)^\ddagger$ | 23.76 | 11.75 | 22.49 | 12.14 | 77.59 | 87.86 | 84.90 | 91.99 | 81.04 | 89.88 |

$\dagger$ used the Swiss-Prot sequence database; $\ddagger$ used NR (non-redundant) database

## 7 Conclusions

We presented new algorithms for inexact matching of the discrete-valued string representations that reduce computational complexity of current algorithms, demonstrate state-of-the-art performance and significantly improved running times. This improvement makes the string kernels with approximate but looser matching a viable alternative for practical tasks of sequence analysis. Our algorithms work with large databases in supervised and semi-supervised settings and scale well in the alphabet size and the number of allowed mismatches. As a consequence, the proposed algorithms can be readily applied to other challenging problems in sequence analysis and mining.

## Footnotes

[1] `http://ranger.uta.edu/~chqding/bioinfo.html`

[2] `http://opihi.cs.uvic.ca/sound/genres`

## References

[1] Jianlin Cheng and Pierre Baldi. A machine learning information retrieval approach to protein fold recognition. *Bioinformatics*, 22(12):1456–1463, June 2006.

[2] Rui Kuang, Eugene Ie, Ke Wang, Kai Wang, Mahira Siddiqi, Yoav Freund, and Christina S. Leslie. Profile-based string kernels for remote homology detection and motif extraction. In *CSB*, pages 152–160, 2004.

[3] Christina S. Leslie, Eleazar Eskin, Jason Weston, and William Stafford Noble. Mismatch string kernels for SVM protein classification. In *NIPS*, pages 1417–1424, 2002.

[4] Sören Sonnenburg, Gunnar Rätsch, and Bernhard Schölkopf. Large scale genomic sequence SVM classifiers. In *ICML '05*, pages 848–855, New York, NY, USA, 2005.

[5] John Shawe-Taylor and Nello Cristianini. *Kernel Methods for Pattern Analysis*. Cambridge University Press, New York, NY, USA, 2004.

[6] Christina Leslie and Rui Kuang. Fast string kernels using inexact matching for protein sequences. *J. Mach. Learn. Res.*, 5:1435–1455, 2004.

[7] Jason Weston, Christina Leslie, Eugene Ie, Dengyong Zhou, Andre Elisseeff, and William Stafford Noble. Semi-supervised protein classification using cluster kernels. *Bioinformatics*, 21(15):3241–3247, 2005.

[8] Vladimir N. Vapnik. *Statistical Learning Theory*. Wiley-Interscience, September 1998.

[9] Christina S. Leslie, Eleazar Eskin, and William Stafford Noble. The spectrum kernel: A string kernel for SVM protein classification. In *Pacific Symposium on Biocomputing*, pages 566–575, 2002.

[10] S. V. N. Vishwanathan and Alex Smola. Fast kernels for string and tree matching. *Advances in Neural Information Processing Systems*, 15, 2002.

[11] M. Gribskov, A.D. McLachlan, and D. Eisenberg. Profile analysis: detection of distantly related proteins. *Proceedings of the National Academy of Sciences*, 84:4355–4358, 1987.

[12] Juho Rousu and John Shawe-Taylor. Efficient computation of gapped substring kernels on large alphabets. *J. Mach. Learn. Res.*, 6:1323–1344, 2005.

[13] Pavel Kuksa, Pai-Hsi Huang, and Vladimir Pavlovic. Fast protein homology and fold detection with sparse spatial sample kernels. In *ICPR 2008*, 2008.

[14] Chris H.Q. Ding and Inna Dubchak. Multi-class protein fold recognition using support vector machines and neural networks. *Bioinformatics*, 17(4):349–358, 2001.

[15] Iain Melvin, Eugene Ie, Jason Weston, William Stafford Noble, and Christina Leslie. Multi-class protein classification using adaptive codes. *J. Mach. Learn. Res.*, 8:1557–1581, 2007.

[16] Tao Li, Mitsunori Ogihara, and Qi Li. A comparative study on content-based music genre classification. In *SIGIR '03*, pages 282–289, New York, NY, USA, 2003. ACM.

[17] Pavel Kuksa, Pai-Hsi Huang, and Vladimir Pavlovic. On the role of local matching for efficient semi-supervised protein sequence classification. In *BIBM*, 2008.

[18] Tommi Jaakkola, Mark Diekhans, and David Haussler. A discriminative framework for detecting remote protein homologies. In *Journal of Computational Biology*, volume 7, pages 95–114, 2000.
